# Zero-Shot Learning with Semantic Output Codes

**Mark Palatucci**
Robotics Institute
Carnegie Mellon University
Pittsburgh, PA 15213
mpalatuc@cs.cmu.edu

**Dean Pomerleau**
Intel Labs
Pittsburgh, PA 15213
dean.a.pomerleau@intel.com

**Geoffrey Hinton**
Computer Science Department
University of Toronto
Toronto, Ontario M5S 3G4, Canada
hinton@cs.toronto.edu

**Tom M. Mitchell**
Machine Learning Department
Carnegie Mellon University
Pittsburgh, PA 15213
tom.mitchell@cs.cmu.edu

## Abstract

We consider the problem of zero-shot learning, where the goal is to learn a classifier $f : X \rightarrow Y$ that must predict novel values of $Y$ that were omitted from the training set. To achieve this, we define the notion of a *semantic output code* classifier (SOC) which utilizes a knowledge base of semantic properties of $Y$ to extrapolate to novel classes. We provide a formalism for this type of classifier and study its theoretical properties in a PAC framework, showing conditions under which the classifier can accurately predict novel classes. As a case study, we build a SOC classifier for a neural decoding task and show that it can often predict words that people are thinking about from functional magnetic resonance images (fMRI) of their neural activity, even without training examples for those words.

## 1   Introduction

Machine learning algorithms have been successfully applied to learning classifiers in many domains such as computer vision, fraud detection, and brain image analysis. Typically, classifiers are trained to approximate a target function $f : X \rightarrow Y$, given a set of labeled training data that includes all possible values for $Y$, and sometimes additional unlabeled training data.

Little research has been performed on *zero-shot learning*, where the possible values for the class variable $Y$ include values that have been *omitted from the training examples*. This is an important problem setting, especially in domains where $Y$ can take on many values, and the cost of obtaining labeled examples for all values is high. One obvious example is computer vision, where there are tens of thousands of objects which we might want a computer to recognize.

Another example is in neural activity decoding, where the goal is to determine the word or object a person is thinking about by observing an image of that person's neural activity. It is intractable to collect neural training images for every possible word in English, so to build a practical neural decoder we must have a way to extrapolate to recognizing words beyond those in the training set.

This problem is similar to the challenges of automatic speech recognition, where it is desirable to recognize words without explicitly including them during classifier training. To achieve vocabulary independence, speech recognition systems typically employ a phoneme-based recognition strategy (Waibel, 1989). Phonemes are the *component parts* which can be combined to construct the words of a language. Speech recognition systems succeed by leveraging a relatively small set of phoneme

recognizers in conjunction with a large knowledge base representing words as combinations of phonemes.

To apply a similar approach to neural activity decoding, we must discover how to infer the component parts of a word's meaning from neural activity. While there is no clear consensus as to how the brain encodes semantic information (Plaut, 2002), there are several proposed representations that might serve as a knowledge base of neural activity, thus enabling a neural decoder to recognize a large set of possible words, even when those words are omitted from a training set.

The general question this paper asks is:

*Given a semantic encoding of a large set of concept classes, can we build a classifier to recognize classes that were omitted from the training set?*

We provide a formal framework for addressing this question and a concrete example for the task of neural activity decoding. We show it is possible to build a classifier that can recognize words a person is thinking about, even without training examples for those particular words.

## 1.1 Related Work

The problem of zero-shot learning has received little attention in the machine learning community. Some work by Larochelle et al. (2008) on *zero-data learning* has shown the ability to predict novel classes of digits that were omitted from a training set. In computer vision, techniques for sharing features across object classes have been investigated (Torralba & Murphy, 2007; Bart & Ullman, 2005) but relatively little work has focused on recognizing entirely novel classes, with the exception of Lampert et al. (2009) predicting visual properties of new objects and Farhadi et al. (2009) using visual property predictions for object recognition.

In the neural imaging community, Kay et al. (2008) has shown the ability to decode (from visual cortex activity) which novel visual scenes a person is viewing from a large set of possible images, but without recognizing the image content per se.

The work most similar to our own is Mitchell (2008). They use semantic features derived from corpus statistics to *generate* a neural activity pattern for any noun in English. In our work, by contrast, we focus on *word decoding*, where given a novel neural image, we wish to predict the word from a large set of possible words. We also consider semantic features that are derived from human labeling in addition to corpus statistics. Further, we introduce a formalism for a zero-shot learner and provide theoretical guarantees on its ability to recognize novel classes omitted from a training set.

## 2 Classification with Semantic Knowledge

In this section we formalize the notion of a zero-shot learner that uses semantic knowledge to extrapolate to novel classes. While a zero-shot learner could take many forms, we present one such model that utilizes an *intermediate set of features* derived from a semantic knowledge base. Intuitively, our goal is to treat each class not as simply an atomic label, but instead represent it using a vector of *semantic features* characterizing a large number of possible classes. Our models will learn the relationship between input data and the semantic features. They will use this learned relationship in a two step prediction procedure to recover the class label for novel input data. Given new input data, the models will predict a set of semantic features corresponding to that input, and then find the class in the knowledge base that best matches that set of predicted features. Significantly, this procedure will even work for input data from a novel class if that class is included in the semantic knowledge base (i.e. even if no input space representation is available for the class, but a feature encoding of it exists in the semantic knowledge base).

### 2.1 Formalism

**Definition 1.** *Semantic Feature Space*
*A semantic feature space of $p$ dimensions is a metric space in which each of the $p$ dimensions encodes the value of a semantic property. These properties may be categorical in nature or may contain real-valued data.*

As an example, consider a semantic space for describing high-level properties of animals. In this example, we'll consider a small space with only $p = 5$ dimensions. Each dimension encodes a binary feature: *is it furry? does it have a tail? can it breathe underwater? is it carnivorous? is it slow moving?* In this semantic feature space, the prototypical concept of *dog* might be represented as the point $\{1, 1, 0, 1, 0\}$.

**Definition 2.** *Semantic Knowledge Base*
*A semantic knowledge base $\mathcal{K}$ of $M$ examples is a collection of pairs $\{f, y\}_{1:M}$ such that $f \in F^p$ is a point in a $p$ dimensional semantic space $F^p$ and $y \in Y$ is a class label from a set $Y$. We assume a one-to-one encoding between class labels and points in the semantic feature space.*

A *knowledge base* of animals would contain the semantic encoding and label for many animals.

**Definition 3.** *Semantic Output Code Classifier*
*A semantic output code classifier $\mathcal{H} : X^d \rightarrow Y$ maps points from some $d$ dimensional raw-input space $X^d$ to a label from a set $Y$ such that $\mathcal{H}$ is the composition of two other functions, $\mathcal{S}$ and $\mathcal{L}$, such that:*

$$
\begin{aligned}
\mathcal{H} &= \mathcal{L}(\mathcal{S}(\cdot)) \\
\mathcal{S} &: X^d \rightarrow F^p \\
\mathcal{L} &: F^p \rightarrow Y
\end{aligned}
$$

This model of a *zero-shot classifier* first maps from a $d$ dimensional raw-input space $X^d$ into a semantic space of $p$ dimensions $F^p$, and then maps this semantic encoding to a class label. For example, we may imagine some raw-input features from a digital image of a dog first mapped into the semantic encoding of a dog described earlier, which is then mapped to the class label *dog*. As a result, our class labels can be thought of as a *semantic output code*, similar in spirit to the error-correcting output codes of Dietterich and Bakiri (1995).

As part of its training input, this classifier is given a set of $N$ examples $\mathcal{D}$ that consists of pairs $\{x, y\}_{1:N}$ such that $x \in X^d$ and $y \in Y$. The classifier is also given a knowledge base $\mathcal{K}$ of $M$ examples that is a collection of pairs $\{f, y\}_{1:M}$ such that $f \in F^p$ and $y \in Y$. Typically, $M >> N$, meaning that data in semantic space is available for many more class labels than in the raw-input space. Thus,

*A semantic output code classifier can be useful when the knowledge base $\mathcal{K}$ covers more of the possible values for $Y$ than are covered by the input data $\mathcal{D}$.*

To learn the mapping $\mathcal{S}$, the classifier first builds a new set of $N$ examples $\{x, f\}_{1:N}$ by replacing each $y$ with the respective semantic encoding $f$ according to its knowledge base $\mathcal{K}$.

The intuition behind using this two-stage process is that the classifier may be able to learn the relationship between the raw-input space and the individual dimensions of the semantic feature space from a relatively small number of training examples in the input space. When a new example is presented, the classifier will make a prediction about its semantic encoding using the learned $\mathcal{S}$ map. Even when a new example belongs to a class that did not appear in the training set $\mathcal{D}$, if the prediction produced by the $\mathcal{S}$ map is close to the true encoding of that class, then the $\mathcal{L}$ map will have a reasonable chance of recovering the correct label. As a concrete example, if the model can predict the object has fur and a tail, it would have a good chance of recovering the class label *dog*, even without having seen images of dogs during training. In short:

*By using a rich semantic encoding of the classes, the classifier may be able to extrapolate and recognize novel classes.*

## 3  Theoretical Analysis

In this section we consider theoretical properties of a *semantic output code classifier* that determine its ability to recognize instances of novel classes. In other words, we will address the question:

*Under what conditions will the semantic output code classifier recognize examples from classes omitted from its training set?*

In answering this question, our goal is to obtain a PAC-style bound: we want to know how much error can be tolerated in the prediction of the semantic properties while still recovering the novel class with high probability. We will then use this error bound to obtain a bound on the number of examples necessary to achieve that level of error in the first stage of the classifier. The idea is that if the first stage $\mathcal{S}(\cdot)$ of the classifier can predict the semantic properties well, then the second stage $\mathcal{L}(\cdot)$ will have a good chance of recovering the correct label for instances from novel classes.

As a first step towards a general theory of zero-shot learning, we will consider one instantiation of a semantic output code classifier. We will assume that semantic features are *binary labels*, the first stage $\mathcal{S}(\cdot)$ is a collection of PAC-learnable linear classifiers (one classifier per feature), and the second stage $\mathcal{L}(\cdot)$ is a 1-nearest neighbor classifier using the *Hamming distance metric*. By making these assumptions, we can leverage existing PAC theory for linear classifiers as well as theory for approximate nearest neighbor search. Much of our nearest-neighbor analysis parallels the work of Ciaccia and Patella (2000).

We first want to bound the amount of error we can tolerate given a prediction of semantic features. To find this bound, we define $\mathcal{F}$ to be the distribution in semantic feature space of points from the knowledge base $\mathcal{K}$. Clearly points (classes) in semantic space may not be equidistant from each other. A point might be far from others, which would allow more room for error in the prediction of semantic features for this point, while maintaining the ability to recover its unique identity (label). Conversely, a point close to others in semantic space will have lower tolerance of error. In short, *the tolerance for error is relative to a particular point in relation to other points in semantic space.*

We next define a prediction $q$ to be the output of the $\mathcal{S}(\cdot)$ map applied to some raw-input example $x \in X^d$. Let $d(q, q')$ be the distance between the prediction $q$ and some other point $q'$ representing a class in the semantic space. We define the relative distribution $R_q$ for a point $q$ as the probability that the distance from $q$ to $q'$ is less than some distance $z$:

$$R_q(z) = \mathbb{P}\left(d(q, q') \leq z\right)$$

This empirical distribution depends on $\mathcal{F}$ and is just the fraction of sampled points from $\mathcal{F}$ that are less than some distance $z$ away from $q$. Using this distribution, we can also define a distribution on the distance to the nearest neighbor of $q$, defined as $\eta_q$:

$$G_q(z) = \mathbb{P}\left(\eta_q \leq z\right)$$

which is given in Ciaccia (2000) as:

$$G_q(z) = 1 - (1 - R_q(z))^n$$

where $n$ is the number of actual points drawn from the distribution $\mathcal{F}$. Now suppose that we define $\tau_q$ to be the distance a prediction $q$ for raw-input example $x$ is from the true semantic encoding of the class to which $x$ belongs. Intuitively, the class we infer for input $x$ is going to be the point closest to prediction $q$, so we want a small probability $\gamma$ that the distance $\tau_q$ to the true class is larger than the distance between $q$ and its nearest neighbor, since that would mean there is a spurious neighbor closer to $q$ in semantic space than the point representing $q$'s true class:

$$\mathbb{P}\left(\tau_q \geq \eta_q\right) \leq \gamma$$

Rearranging we can put this in terms of the distribution $G_q$ and then can solve for $\tau_q$:

$$
\begin{aligned}
\mathbb{P}\left(\eta_q \leq \tau_q\right) &\leq \gamma \\
G_q(\tau_q) &\leq \gamma
\end{aligned}
$$

If $G_q(\cdot)$ were invertible, we could immediately recover the value $\tau_q$ for a desired $\gamma$. For some distributions, $G_q(\cdot)$ may not be a 1-to-1 function, so there may not be an inverse. But $G_q(\cdot)$ will never decrease since it is a cumulative distribution function. We will therefore define a function $G_q^{-1}$ such that: $G_q^{-1}(\gamma) = \operatorname{argmax}_{\tau_q}\left[G_q(\tau_q) \leq \gamma\right]$.

So using nearest neighbor for $\mathcal{L}(\cdot)$, if $\tau_q \leq G_q^{-1}(\gamma)$, then we will recover the correct class with at least $1 - \gamma$ probability. To ensure that we achieve this error bound, we need to make sure the total error of $\mathcal{S}(\cdot)$ is less than $G_q^{-1}(\gamma)$ which we define as $\tau_q^{max}$. We assume in this analysis that we have $p$ binary semantic features and a Hamming distance metric, so $\tau_q^{max}$ defines the total

number of mistakes we can make predicting the binary features. Note with our assumptions, each semantic feature is PAC-learnable using a linear classifier from a $d$ dimensional raw input space. To simplify the analysis, we will treat each of the $p$ semantic features as independently learned. By the PAC assumption, the true error (i.e. probability of the classifier making a mistake) of each of the $p$ learned hypotheses is $\epsilon$, then the expected number of mistakes over the $p$ semantic features will be $\tau_q^{max}$ if we set $\epsilon = \tau_q^{max}/p$. Further, the probability of making at most $\tau_q^{max}$ mistakes is given by the binomial distribution: $\text{BinoCDF}(\tau_q^{max}; p, \tau_q^{max}/p)$

We can obtain the desired error rate for each hypothesis by utilizing the standard PAC bound for VC-dimension[1] (Mitchell, 1997). To obtain a hypothesis with $(1 - \delta)$ probability that has true error at most $\epsilon = \tau_q^{max}/p = G^{-1}(\gamma)/p$, then the classifier requires a number of examples $M_{q,\delta}$:

$$M_{q,\delta} \quad \geq \quad \frac{p}{\tau_q^{max}}\left[4\log(2/\delta) + 8(d + 1)\log(13p/\tau_q^{max})\right] \tag{1}$$

If each of the $p$ classifiers (feature predictors) is learned with this many examples, then with probability $(1 - \delta)^p$, *all* feature predictors will achieve the desired error rate. But note that this is only the probability of achieving $p$ hypotheses with the desired true error rate. The binomial CDF yields the probability of making at most $\tau_q^{max}$ mistakes total, and the $(1 - \gamma)$ term above specifies the probability of recovering the true class if a maximum of this many mistakes were made. Therefore, there are three probabilistic events required for the semantic output code classifier to predict a novel class and the total (joint) probability of these events is:

$\mathbb{P}\,(\text{there are } p \text{ feature predictors with true error} \leq \tau_q^{max}/p)\;\cdot$

$\mathbb{P}\,(\text{at most } \tau_q^{max} \text{ mistakes made} \mid \text{there are } p \text{ feature predictors with true error} \leq \tau_q^{max}/p\,)\cdot$

$\mathbb{P}\,(\text{recovering true class} \mid \text{at most } \tau_q^{max} \text{ mistakes made})$

and since $\tau_q^{max} = G_q^{-1}(\gamma)$, the total probability is given by:

$$(1 - \delta)^p \cdot \text{BinoCDF}(G_q^{-1}(\gamma); p, G_q^{-1}(\gamma)/p) \cdot (1 - \gamma) \tag{2}$$

In summary, given desired error parameters $(1-\gamma)$ and $(1-\delta)$ for the two classifier stages, Equation 2 provides the total probability of correctly predicting a novel class. Given the value for $\gamma$ we can compute the $\epsilon$ necessary for each feature predictor. We are guaranteed to obtain the total probability if the feature predictors were trained with $M_{q,\delta}$ raw-input examples as specified in Equation 1.

*To our knowledge, Equations 1 and 2 specify the first formal guarantee that provides conditions under which a classifier can predict novel classes.*

## 4 Case Study: Neural Decoding of Novel Thoughts

In this section we empirically evaluate a semantic output code classifier on a neural decoding task. The objective is to decode novel words a person is thinking about from fMRI images of the person's neural activity, without including example fMRI images of those words during training.

### 4.1 Datasets

We utilized the same fMRI dataset from Mitchell (2008). This dataset contains the neural activity observed from nine human participants while viewing 60 different concrete words (5 examples from 12 different categories). Some examples include animals: `bear, dog, cat, cow, horse` and vehicles: `truck, car, train, airplane, bicycle`. Each participant was shown a word and a small line drawing of the concrete object the word represents. The participants were asked to think about the properties of these objects for several seconds while images of their brain activity were recorded.

Each image measures the neural activity at roughly 20,000 locations (i.e. voxels) in the brain. Six fMRI scans were taken for each word. We used the same time-averaging described in Mitchell (2008) to create a single average brain activity pattern for each of the 60 words, for each participant.

In the language of the semantic output code classifier, this dataset represents the collection $\mathcal{D}$ of raw-input space examples.

We also collected two semantic knowledge bases for these 60 words. In the first semantic knowledge base, `corpus5000`, each word is represented as a co-occurrence vector with the 5000 most frequent words from the Google Trillion-Word-Corpus[2].

The second semantic knowledge base, `human218`, was created using the Mechanical Turk human computation service from Amazon.com. There were 218 semantic features collected for the 60 words, and the questions were selected to reflect psychological conjectures about neural activity encoding. For example, the questions related to size, shape, surface properties, and typical usage. Example questions include *is it manmade?* and *can you hold it?*. Users of the Mechanical Turk service answered these questions for each word on a scale of 1 to 5 (definitely not to definitely yes).

## 4.2 Model

In our experiments, we use *multiple output linear regression* to learn the $\mathcal{S}(\cdot)$ map of the semantic output code classifier. Let $\mathbf{X} \in \Re^{N*d}$ be a training set of fMRI examples where each row is the image for a particular word and $d$ is the number of dimensions of the fMRI image. During training, we use the *voxel-stability-criterion* that does not use the class labels described in Mitchell (2008) to reduce $d$ from about 20,000 voxels to 500. Let $\mathbf{Y} \in \Re^{N*p}$ be a *matrix* of semantic features for those words (obtained from the knowledge base $\mathcal{K}$) where $p$ is the number of semantic features for that word (e.g. 218 for the `human218` knowledge base). We learn a matrix of weights $\widehat{\mathbf{W}} \in \Re^{d*p}$. In this model, each output is treated independently, so we can solve all of them quickly in one matrix operation (even with thousands of semantic features):

$$\widehat{\mathbf{W}} = (\mathbf{X}^T\mathbf{X} + \lambda\mathbf{I})^{-1}\mathbf{X}^T\mathbf{Y} \tag{3}$$

where $\mathbf{I}$ is the identity matrix and $\lambda$ is a regularization parameter chosen automatically using the cross-validation scoring function (Hastie et al., 2001)[3]. Given a novel fMRI image $\mathbf{x}$, we can obtain a prediction $\widehat{\mathbf{f}}$ of the semantic features for this image by multiplying the image by the weights: $\widehat{\mathbf{f}} = \mathbf{x} \cdot \widehat{\mathbf{W}}$

For the second stage of the semantic output code classifier, $\mathcal{L}(\cdot)$, we simply use a 1-nearest neighbor classifier. In other words, $\mathcal{L}(\widehat{\mathbf{f}})$ will take the prediction of features and return the closest point in a given knowledge base according the Euclidean distance ($L_2$) metric.

## 4.3 Experiments

Using the model and datasets described above, we now pose and answer three important questions.

*1. Can we build a classifier to discriminate between two classes, where neither class appeared in the training set?*

To answer this question, we performed a *leave-two-out-cross-validation*. Specifically, we trained the model in Equation 3 to learn the mapping between 58 fMRI images and the semantic features for their respective words. For the first held out image, we applied the learned weight matrix to obtain a prediction of the semantic features, and then we used a 1-nearest neighbor classifier to compare the vector of predictions to the true semantic encodings of the *two held-out words*. The label was chosen by selecting the word with the encoding closest to the prediction for the fMRI image. We then performed the same test using the second held-out image. Thus, for each iteration of the cross-validation, two separate comparisons were made. This process was repeated for all $\binom{60}{2} = 1,770$ possible leave-two-out combinations leading to 3,540 total comparisons.

Table 1 shows the results for two different semantic feature encodings. We see that the `human218` semantic features significantly outperformed the `corpus5000` features, with mean accuracies over the nine participants of 80.9% and 69.7% respectively. But for both feature sets, we see that *it is possible to discriminate between two novel classes for each of the nine participants.*

Table 1: Percent accuracies for leave-two-out-cross-validation for 9 fMRI participants (labeled P1-P9). The values represent classifier percentage accuracy over 3,540 trials when discriminating between two fMRI images, both of which were omitted from the training set.

|  | P1 | P2 | P3 | P4 | P5 | P6 | P7 | P8 | P9 | Mean |
|---|---|---|---|---|---|---|---|---|---|---|
| corpus5000 | 79.6 | 67.0 | 69.5 | 56.2 | 77.7 | 65.5 | 71.2 | 72.9 | 67.9 | **69.7** |
| human218 | 90.3 | 82.9 | 86.6 | 71.9 | 89.5 | 75.3 | 78.0 | 77.7 | 76.2 | **80.9** |

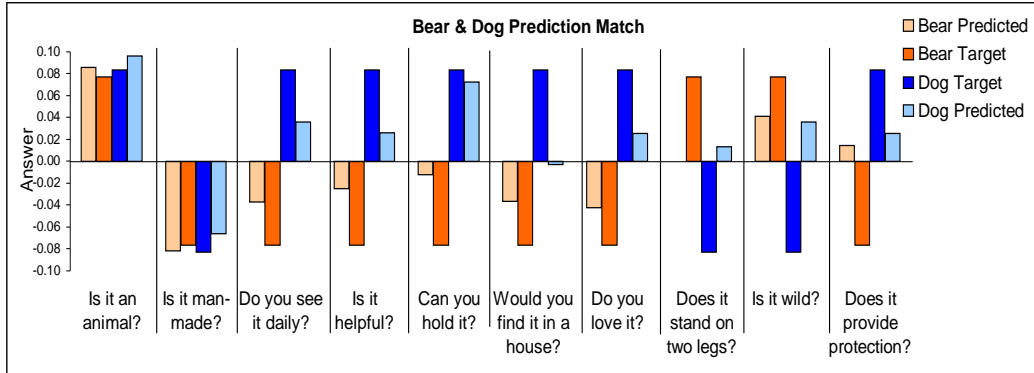

Figure 1: Ten semantic features from the human218 knowledge base for the words *bear* and *dog*. The true encoding is shown along with the predicted encoding when fMRI images for bear and dog were left out of the training set.

*2. How is the classifier able to discriminate between closely related novel classes?*

Figure 1 shows ten semantic questions (features) from the human218 dataset. The graph shows the true values along with the predicted feature values for both *bear* and *dog* when trained on the other 58 words. We see the model is able to learn to predict many of the key features that bears and dogs have in common such as *is it an animal?* as well as those that differentiate between the two, such as *do you see it daily?* and *can you hold it?* For both of these novel words, the features predicted from the neural data were closest to the true word.

*3. Can we decode the word from a large set of possible words?*

Given the success of the semantic output code classifier at discriminating between the brain images for two novel words, we now consider the much harder problem of discriminating a novel word from a large set of candidate words. To test this ability, we performed a *leave-one-out-cross-validation*, where we trained using Equation 3 on images and semantic features for 59 words. We then predicted the features for the held-out image of the 60th word, and then performed a 1-nearest neighbor classification in a large set of candidate words.

We tested two different word sets. The first was mri60 which is the collection of all 60 concrete nouns for which we collected fMRI data, including the 59 training words and the single held out word. The second set was noun940, a collection of 940 English nouns with high familiarity, concreteness and imagineability, compiled from Wilson (1988) and Snodgrass (1980). For this set of words, we added the true held-out word to the set of 940 on each cross-validation iteration. We performed this experiment using both the corpus5000 and human218 feature sets. The rank accuracy results (over 60 cross-validation iterations) of the four experiments are shown in Figure 2.

The human218 features again significantly outperform corpus5000 on both mean and median rank accuracy measures, and both feature sets perform well above chance. On 12 of 540 total presentations of the mri60 words (60 presentations for each of nine participants), the human218 features *predicted the single held-out word above all 59 other words in its training set*. While just a bit above chance level (9/540), the fact that the model *ever* chooses the held-out word over all the training words is noteworthy since the model is undoubtedly biased towards predicting feature values similar to the words on which it was trained. On the noun940 words, the model predicted the correct word from the set of 941 alternatives *a total of 26 times* for the human218 features and 22 times for the corpus5000 features. For some subjects, the model correctly picked the right

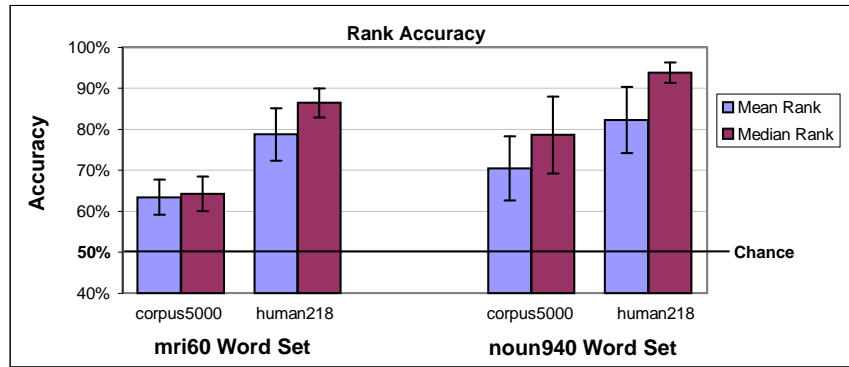

Figure 2: The mean and median rank accuracies across nine participants for two different semantic feature sets. Both the original 60 fMRI words and a set of 940 nouns were considered.

Table 2: The top five predicted words for a novel fMRI image taken for the word in bold (all fMRI images taken from participant P1). The number in the parentheses contains the rank of the correct word selected from 941 concrete nouns in English.

| **Bear** | **Foot** | **Screwdriver** | **Train** | **Truck** | **Celery** | **House** | **Pants** |
|---|---|---|---|---|---|---|---|
| (1) | (1) | (1) | (1) | (2) | (5) | (6) | (21) |
| *bear* | *foot* | *screwdriver* | *train* | jeep | beet | supermarket | clothing |
| fox | feet | pin | jet | *truck* | artichoke | hotel | vest |
| wolf | ankle | nail | jail | minivan | grape | theater | t-shirt |
| yak | knee | wrench | factory | bus | cabbage | school | clothes |
| gorilla | face | dagger | bus | sedan | *celery* | factory | panties |

word from the set of 941 more than 10% of the time. The chance accuracy of predicting a word correctly is only 0.1%, meaning we would expect less than one correct prediction across all 540 presentations.

As Figure 2 shows, the median rank accuracies are often significantly higher than the mean rank accuracies. Using the human218 features on the noun940 words, the median rank accuracy is above 90% for each participant while the mean is typically about 10% lower. This is due to the fact that several words are consistently predicted poorly. The prediction of words in the categories *animals, body parts, foods, tools, and vehicles* typically perform well, while the words in the categories *furniture, man-made items, and insects* often perform poorly.

Even when the correct word is not the closest match, the words that best match the predicted features are often very similar to the held-out word. Table 2 shows the top five predicted words for eight different held-out fMRI images for participant P1 (i.e. the 5 closest words in the set of 941 to the predicted vector of semantic features).

## 5   Conclusion

We presented a formalism for a *zero-shot learning* algorithm known as the *semantic output code classifier*. This classifier can predict novel classes that were omitted from a training set by leveraging a semantic knowledge base that encodes features common to both the novel classes and the training set. We also proved the first formal guarantee that shows conditions under which this classifier will predict novel classes.

We demonstrated this semantic output code classifier on the task of neural decoding using semantic knowledge bases derived from both human labeling and corpus statistics. We showed this classifier can predict the word a person is thinking about from a recorded fMRI image of that person's neural activity with accuracy much higher than chance, even when training examples for that particular word were omitted from the training set and the classifier was forced to pick the word from among nearly 1,000 alternatives.

We have shown that training images of brain activity are *not* required for every word we would like a classifier to recognize. These results significantly advance the state-of-the-art in neural decoding and are a promising step towards a large vocabulary brain-computer interface.

## Footnotes

[1]The VC dimension of linear classifiers in $d$ dimensions is $d + 1$

[2]Vectors are normalized to unit length and do not include 100 stop words like *a, the, is*

[3]We compute the cross-validation score for each task and choose the parameter that minimizes the average loss across all output tasks.

# References

Bart, E., & Ullman, S. (2005). Cross-generalization: learning novel classes from a single example by feature replacement. *Computer Vision and Pattern Recognition, 2005. CVPR 2005. IEEE Computer Society Conference on*, *1*, 672–679 vol. 1.

Ciaccia, P., & Patella, M. (2000). PAC nearest neighbor queries: Approximate and controlled search in high-dimensional and metric spaces. *Data Engineering, International Conference on*, 244.

Dietterich, T. G., & Bakiri, G. (1995). Solving multiclass learning problems via error-correcting output codes. *Journal of Artificial Intelligence Research*.

Farhadi, A., Endres, I., Hoiem, D., & Forsyth, D. (2009). Describing objects by their attributes. *Proceedings of the IEEE Computer Society Conference on Computer Vision and Pattern Recognition (CVPR)*.

Hastie, T., Tibshirani, R., & Friedman, J. H. (2001). *The elements of statistical learning*. Springer.

Kay, K. N., Naselaris, T., Prenger, R. J., & Gallant, J. L. (2008). Identifying natural images from human brain activity. *Nature*, *452*, 352–355.

Lampert, C. H., Nickisch, H., & Harmeling, S. (2009). Learning to detect unseen object classes by between-class attribute transfer. *Proceedings of the IEEE Computer Society Conference on Computer Vision and Pattern Recognition (CVPR)*.

Larochelle, H., Erhan, D., & Bengio, Y. (2008). Zero-data learning of new tasks. *AAAI Conference on Artificial Intelligence*.

Mitchell, T., et al. (2008). Predicting human brain activity associated with the meanings of nouns. *Science*, *320*, 1191–1195.

Mitchell, T. M. (1997). *Machine learning*. New York: McGraw-Hill.

Mitchell, T. M., Hutchinson, R., Niculescu, R. S., Pereira, F., Wang, X., Just, M., & Newman, S. (2004). Learning to decode cognitive states from brain images. *Machine Learning*, *57*, 145–175.

Plaut, D. C. (2002). Graded modality-specific specialization in semantics: A computational account of optic aphasia. *Cognitive Neuropsychology*, *19*, 603–639.

Snodgrass, J., & Vanderwart, M. (1980). A standardized set of 260 pictures: Norms for name agreement, image agreement, familiarity and visual complexity. *Journal of Experimental Psychology: Human Learning and Memory*, 174–215.

Torralba, A., & Murphy, K. P. (2007). Sharing visual features for multiclass and multiview object detection. *IEEE Trans. Pattern Anal. Mach. Intell.*, *29*, 854–869.

van der Maaten, L., & Hinton, G. (2008). Visualizing data using t-SNE. *Journal of Machine Learning Research*, *9(Nov)*, 2579–2605.

Waibel, A. (1989). Modular construction of time-delay neural networks for speech recognition. *Neural Computation*, *1*, 39–46.

Wilson, M. (1988). The MRC psycholinguistic database: Machine readable dictionary, version 2. *Behavioral Research Methods*, 6–11.

